# Neural Control for Rolling Mills: Incorporating Domain Theories to Overcome Data Deficiency

**Martin Röscheisen**
Computer Science Dept.
Munich Technical University
8 Munich 40, FRG

**Reimar Hofmann**
Computer Science Dept.
Edinburgh University
Edinburgh, EH89A, UK

**Volker Tresp**
Corporate R & D
Siemens AG
8 Munich 83, FRG

## Abstract

In a Bayesian framework, we give a principled account of how domain-specific prior knowledge such as imperfect analytic domain theories can be optimally incorporated into networks of locally-tuned units: by choosing a specific architecture and by applying a specific training regimen. Our method proved successful in overcoming the data deficiency problem in a large-scale application to devise a neural control for a hot line rolling mill. It achieves in this application significantly higher accuracy than optimally-tuned standard algorithms such as sigmoidal backpropagation, and outperforms the state-of-the-art solution.

## 1   INTRODUCTION

Learning in connectionist networks typically requires many training examples and relies more or less explicitly on some kind of *syntactic* preference bias such as "minimal architecture" (Rumelhart, 1988; Le Cun *et al.*, 1990; Weigend, 1991; inter alia) or a smoothness constraint operator (Poggio *et al.*, 1990), but does not make use of explicit representations of domain-specific prior knowledge. If training data is deficient, learning a functional mapping inductively may no longer be feasible, whereas this may still be the case when guided by domain knowledge. Controlling a rolling mill is an example of a large-scale real-world application where training data is very scarce and noisy, yet there exist much refined, though still very approximate, analytic models that have been applied for the past decades and embody many years of experience in this particular domain. Much in the spirit of Explanation-

Based Learning (see, for example, Mitchell *et al.*, 1986; Minton *et al.*, 1986), where domain knowledge is applied to get valid generalizations from only a few training examples, we consider an analytic model as an imperfect domain theory from which the training data is "explained" (see also Scott *et al.*, 1991; Bergadano *et al.*, 1990; Tecuci *et al.*, 1990). Using a Bayesian framework, we consider in Section 2 the optimal response of networks in the presence of noise on their input, and derive, in Section 2.1, a familiar localized network architecture (Moody *et al.*, 1989, 1990). In Section 2.2, we show how domain knowledge can be readily incorporated into this localized network by applying a specific training regimen. These results were applied as part of a project to devise a neural control for a hot line rolling mill, and, in Section 3, we describe experimental results which indicate that incorporating domain theories can be indispensable for connectionist networks to be successful in difficult engineering domains. (See also references for one of our more detailed papers.)

# 2    THEORETICAL FOUNDATION

## 2.1    NETWORK ARCHITECTURE

We apply a Bayesian framework to systems where the training data is assumed to be generated from the true model $f$, which itself is considered to be derived from a domain theory $b$ that is represented as a function. Since the measurements in our application are very noisy and clustered, we took this as the paradigm case, and assume the actual input $X \in \mathbb{R}^d$ to be a noisy version of one of a small number ($N$) of prototypical input vectors $\vec{t}_1, \ldots, \vec{t}_N \in \mathbb{R}^d$ where the noise is additive with covariance matrix $\Sigma$. The corresponding true output values $f(\vec{t}_1), \ldots, f(\vec{t}_N) \in \mathbb{R}$ are assumed to be distributed around the values suggested by the domain theory, $b(\vec{t}_1), \ldots, b(\vec{t}_N)$ (variance $\sigma_{prior}^2$). Thus, each point in the training data $D := \{(\vec{x}_i, y_i); \ i = 1, \ldots, M\}$ is considered to be generated as follows: $\vec{x}_i$ is obtained by selecting one of the $\vec{t}_k$ and adding zero-mean noise with covariance $\Sigma$, and $y_i$ is generated by adding Gaussian zero-mean noise with variance $\sigma_{data}^2$ to $f(\vec{t}_k)$.[1] We determine the system's response $O(\vec{x})$ to an input $\vec{x}$ to be optimal with respect to the expectation of the squared error (MMSE-estimate):

$$O(\vec{x}) := \underset{o(\vec{x})}{argmin} \ \mathcal{E}((f(T_{true}) - o(\vec{x}))^2).$$

The expectation is given by $\sum_{i=1}^{N} P(T_{true} = \vec{t}_i | X = \vec{x}) \cdot (f(\vec{t}_i) - o(\vec{x}))^2$. Bayes' Theorem states that $P(T_{true} = \vec{t}_i | X = \vec{x}) = p(X = \vec{x} | T_{true} = \vec{t}_i) \cdot P(T_{true} = \vec{t}_i) / p(X = \vec{x})$. Under the assumption that all $\vec{t}_i$ are equally likely, simplifying the derivative of the expectation yields

$$O(\vec{x}) = \frac{\sum_{i=1}^{N} p(X = \vec{x} | T_{true} = \vec{t}_i) \cdot C_i}{\sum_{i=1}^{N} p(X = \vec{x} | T_{true} = \vec{t}_i)}$$

where $C_i$ equals $\mathcal{E}(f(\vec{t_i})|D)$, *i.e.* the expected value of $f(\vec{t_i})$ given that the training data is exactly D. Assuming the input noise to be Gaussian and $\Sigma$, unless otherwise noted, to be diagonal, $\Sigma = (\delta_{ij}\sigma_i^2)_{1 \le i,j \le d}$, the probability density of $X$ under the assumption that $T_{true}$ equals $\vec{t_k}$ is given by

$$p(X = \vec{x}|T_{true} = \vec{t_k}) = \frac{1}{(2\pi)^{d/2} \cdot |\Sigma|^{1/2}} \exp\left[-\frac{1}{2}(\vec{x} - \vec{t_k})^t \, \Sigma^{-1} \, (\vec{x} - \vec{t_k})\right]$$

where $|.|$ is the determinant. The optimal response to an input $\vec{x}$ can now be written as

$$O(x) = \frac{\sum_{i=1}^{N} \exp\left[-\frac{1}{2}(\vec{x} - \vec{t_i})^t \, \Sigma^{-1} \, (\vec{x} - \vec{t_i})\right] \cdot C_i}{\sum_{i=1}^{N} \exp\left[-\frac{1}{2}(\vec{x} - \vec{t_i})^t \, \Sigma^{-1} \, (\vec{x} - \vec{t_i})\right]}. \tag{1}$$

Equation 1 corresponds to a network architecture with $N$ Gaussian Basis Functions (GBFs) centered at $\vec{t_k}$, $k = 1, \ldots, N$, each of which has a width $\sigma_i$, $i = 1, \ldots, d$, along the $i$-th dimension, and an output weight $C_k$. This architecture is known to give smooth function approximations (Poggio *et al.*, 1990; see also Platt, 1990), and the normalized response function (partitioning-to-one) was noted earlier in studies by Moody *et al.* (1988, 1989, 1990) to be beneficial to network performance. Carving up an input space into hyperquadrics (typically hyperellipsoids or just hyperspheres) in this way suffers in practice from the severe drawback that as soon as the dimensionality of the input is higher, it becomes less feasible to cover the whole space with units of only local relevance ("curse of dimensionality"). The normalized response function has an essentially space-filling effect, and fewer units have to be allocated while, at the same time, most of the locality properties can be preserved such that efficient ball tree data structures (Omohundro, 1991) can still be used. If the distances between the centers are large with respect to their widths, the nearest-neighbor rule is recovered. With decreasing distances, the output of the network changes more smoothly between the centers.

## 2.2  TRAINING REGIMEN

The output weights $C_i$ are given by

$$C_i = \mathcal{E}(f(\vec{t_i})|D) = \int_{-\infty}^{\infty} z \cdot p(f(\vec{t_i}) = z|D)\, dz \,.$$

Bayes' Theorem states that $p(f(\vec{t_i}) = z|D) = p(D|f(\vec{t_i}) = z) \cdot p(f(\vec{t_i}) = z) \,/\, p(D)$. Let $M(i)$ denote the set of indices $j$ of the training data points $(x_j, y_j)$ that were generated by adding noise to $(\vec{t_i}, f(\vec{t_i}))$, *i.e.* the points that "originated" from $\vec{t_i}$. Note that it is not known *a priori* which indices a set $M(i)$ contains; only posterior probabilities can be given. By applying Bayes' Theorem and by assuming the independence between different locations $\vec{t_i}$, the coefficients $C_i$ can be written as[2]

$$C_i = \int_{-\infty}^{\infty} z \cdot \frac{\prod_{m \in M(i)} \exp\left[-\frac{1}{2}\frac{(z-y_m)^2}{\sigma_{data}^2}\right] \cdot \exp\left[-\frac{1}{2}\frac{(z-b(\vec{t_i}))^2}{\sigma_{prior}^2}\right]}{\int_{-\infty}^{\infty} \prod_{m \in M(i)} \exp\left[-\frac{1}{2}\frac{(v-y_m)^2}{\sigma_{data}^2}\right] \cdot \exp\left[-\frac{1}{2}\frac{(v-b(\vec{t_i}))^2}{\sigma_{prior}^2}\right]\, dv}\, dz.$$

It can be easily shown that this simplifies to

$$C_i = \frac{\sum_{m \in M(i)} y_m + k \cdot b(\vec{t_i})}{|M(i)| + k} \tag{2}$$

where $k = \sigma^2_{data}/\sigma^2_{prior}$ and $|.|$ denotes the cardinality operator. In accordance with intuition, the coefficients $C_i$ turn out to be a weighted mean between the value suggested by the domain theory $b$ and the training data values which originated from $\vec{t_i}$. The weighting factor $k/(|M(i)| + k)$ reflects the relative reliability of the two sources of information, the empirical data and the prior knowledge.

Define $S_i$ as $S_i = (C_i - b(\vec{t_k})) \cdot k + \sum_{m \in M(i)} (C_i - y_m)$. Clearly, if $|S_i|$ is minimized to 0, then $C_i$ reaches exactly the optimal value as it is given by equation 2. An adaptive solution to this is to update $C_i$ according to $\dot{C_i} = -\gamma \cdot S_i$. Since the membership distribution for $M(i)$ is not known *a priori*, we approximate it using a posterior estimate of the probability $p(m \in M(i)|\vec{x}_m)$ that $m$ is in $M(i)$ given that $\vec{x}_m$ was generated by some center $\vec{t_k}$, which is

$$p(m \in M(i)|\vec{x}_m) = \frac{p(X = \vec{x}_m|T_{true} = \vec{t_i})}{\sum_{k=1}^M p(X = \vec{x}_m|T_{true} = \vec{t_k})}.$$

$p(X = \vec{x}_m|T_{true} = \vec{t_i})$ is the activation $act_i$ of the $i$-th center, when the network is presented with input $\vec{x}_m$. Substituting the equation in the sum of $S_i$ leads to the following training regimen: Using stochastic sample-by-sample learning, we present in each training step with probability $1 - \lambda$ a data point $y_j$, and with probability $\lambda$ a point $b(\vec{t_k})$ that is generated from the domain theory, where $\lambda$ is given by

$$\lambda := \frac{k \cdot N}{k \cdot N + M}. \tag{3}$$

(Recall that $M$ is the total number of data points, and N is the number of centers.) $\lambda$ varies from 0 (the data is far more reliable than the prior knowledge) to 1 (the data is unreliable in comparison with the prior knowledge). Thus, the change of $C_i$ after each presentation is proportional to the error times the normalized activation of the $i$-th center, $act_i / \sum_{k=1}^N act_k$.

The optimal positions for the centers $\vec{t_i}$ are not known in advance, and we therefore perform standard LMS gradient descent on $\vec{t_i}$, and on the widths $\sigma_i$. The weight updates in a learning step are given by a discretization of the following dynamic equations (i=1,...,N; j=1,...,d):

$$\dot{t}_{ij} = 2\gamma \cdot \Delta \cdot act_i \cdot \frac{C_i - O(\vec{x})}{\sum_{k=1}^N act_k} \cdot \frac{1}{\sigma^2_{ij}} \cdot (x_j - t_{ij})$$

$$\left(\frac{\dot{1}}{\sigma^2_{ij}}\right) = -\gamma \cdot \Delta \cdot act_i \cdot \frac{C_i - O(\vec{x})}{\sum_{k=1}^N act_k} \cdot (x_j - t_{ij})^2$$

where $\Delta$ is the interpolation error, $act_i$ is the (forward-computed) activity of the the $i$-th center, and $t_{ij}$ and $x_j$ are the $j$-th component of $\vec{t_i}$ and $\vec{x}$ respectively.

# 3    APPLICATION TO ROLLING MILL CONTROL

## 3.1    THE PROBLEM

In integrated steelworks, the finishing train of the hot line rolling mill transforms preprocessed steel from a casting successively into a homogeneously rolled steel-plate. Controlling this process is a notoriously hard problem: The underlying physical principles are only roughly known. The values of the control parameters depend on a large number of entities, and have to be determined from measurements that are very noisy, strongly clustered, "expensive," and scarce.[3] On the other hand, reliability and precision are at a premium. Unreasonable predictions have to be avoided under any circumstances, even in regions where no training data is available, and, by contract, an extremely high precision is required: the rolling tolerance has to be guaranteed to be less than typically $20\mu$m, which is substantial, particularly in the light of the fact that the steel construction that holds the rolls itself expands for several millimeters under a rolling pressure of typically several thousands of tons. The considerable economic interest in improving adaptation methods in rolling mills derives from the fact that lower rolling tolerances are indispensable for the supplied industry, yet it has proven difficult to remain operational within the guaranteed bounds under these constraints.

The control problem consists of determining a reduction schedule that specifies for each pair of rolls their initial distance such that after the final roll pair the desired thickness of the steel-plate (the actual feedback) is achieved. This reinforcement problem can be reduced to a less complex approximation problem of predicting the rolling force that is created at each pair of rolls, since this force can directly and precisely be correlated to the reduction in thickness at a roll pair by conventional means. Our task was therefore to predict the rolling force on the basis of nine input variables like temperature and rolling speed, such that a subsequent conventional high-precision control can quickly reach the guaranteed rolling tolerance before much of a plate is lost.

The state-of-the-art solution to this problem is a parameterized analytic model that considers nine physical entities as input and makes use of a huge number of tabulated coefficients that are adapted separately for each material and each thickness class. The solution is known to give only approximate predictions about the actual force, and although the on-line corrections by the high-precision control are generally sufficient to reach the rolling tolerance, this process necessarily takes more time, the worse the prediction is—resulting in a waste of more of the beginning of a steel-plate. Furthermore, any improvement in the adaptation techniques will also shorten the initialization process for a rolling mill, which currently takes several months because of the poor generalization abilities of the applied method to other thickness classes or steel qualities.

The data for our simulations was drawn from a rolling mill that was being installed at the time of our experiments. It included measurements for around 200 different steel qualities; only a few qualities were represented more than 100 times.

## 3.2   EXPERIMENTAL RESULTS

According to the results in Section 2, a network of the specified localized architecture was trained with data (artificially) generated from the domain theory and data derived from on-line measurements. The remaining design considerations for architecture selection were based on the extent to which a network had the capacity to represent an instantiation of the analytic model (our domain theory):

Table 1 shows the approximation error of partitioning-to-one architectures with different degrees of freedom on their centers' widths. The variances of the GBFs were either all equal and not adapted (GBFs with constant widths), or adapted individually for all centers (GBFs with spherical adaptation), or adapted individually for all centers and every input dimension—leading to axially oriented hyperellipsoids (GBFs with ellipsoidal adaptation). Networks with "full hyperquadric" GBFs, for

| Method | Normalized Error Squares $[10^{-2}]$ | Maximum Error $[10^{-2}]$ |
|---|---|---|
| GBFs with partitioning | | |
| constant widths | 0.40 | 2.1 |
| spherical adaptation | 0.18 | 1.7 |
| ellipsoidal " | 0.096 | 0.41 |
| GBFs no partitioning | 0.85 | 5.3 |
| MLP | 0.38 | 3.4 |

Table 1: Approximation of an instantiation of the domain theory: localized architectures (GBFs) and a network with sigmoidal hidden units (MLP).

which the covariance matrix is no longer diagonal, were also tested, but performed clearly worse, apparently due to too many degrees of freedom. The table shows that the networks with "ellipsoidal" GBFs performed best. Convergence time of this type of network was also found to be superior. The table also gives the comparative numbers for two other architectures: GBFs without normalized response function achieved significantly lower accuracy—even if they had far more centers (performance is given for a net with 81 centers)—than those with partitioning and only 16 centers. Using up to 200 million sample presentations, sigmoidal networks trained with standard backpropagation (Rumelhart *et al.*, 1986) achieved a yet lower level—despite the use of weight-elimination (Le Cun, 1990), and an analysis of the data's eigenvalue spectrum to optimize the learning rate (see also Le Cun, 1991). The indicated numbers are for networks with optimized numbers of hidden units.

The value for $\lambda$ was determined according to equation 3 in Section 2.2 as $\lambda = 0.8$; the noise in our application could be easily estimated, since there are multiple measurements for each input point available and the reliability of the domain theory is known. Applying the described training regimen to the GBF-architecture with ellipsoidal adaptation led to promising results:

Figure 1 shows the points in a "slice" through a specific point in the input space: the measurements, the force as it is predicted by the analytic model and the network. It can be seen that the net exhibits fail-safe behavior: it sticks closely to the analytic model in regions where no data is available. If data points are available and suggest

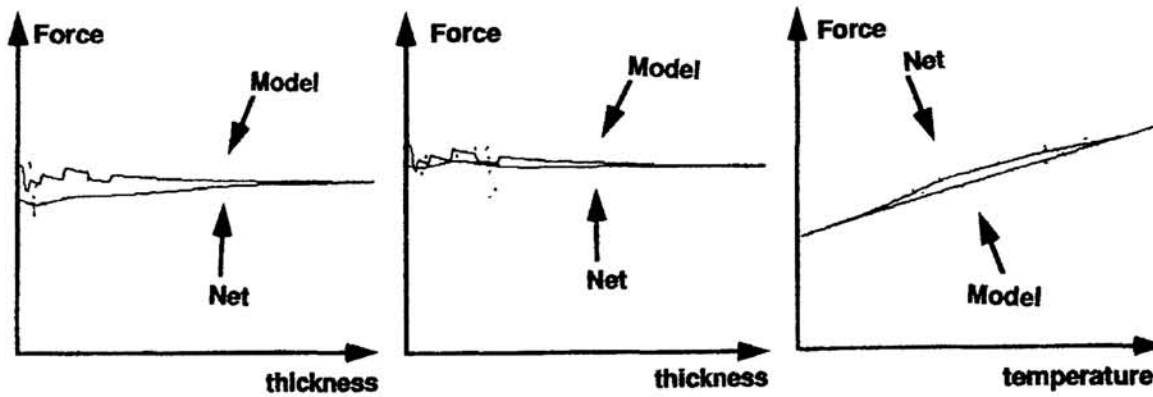

Figure 1: Prediction of the rolling force by the state-of-the-art model, by the neural network, and the measured data points as a function of the input 'sheet thickness,' and 'temperature.'

| Method | Percent of Improvement on Trained Samples | Percent of Improvement at Generalization |
|---|---|---|
| Gaussian Units    $\lambda = 0.8$ | 18 | 16 |
| Gaussian Units    $\lambda = 0.4$<br>MLP | 41<br>3.9 | 14<br>3.1 |

Table 2: Relative improvement of the neural network solutions with respect to the state-of-the-art model: on the training data and on the cross-validation set.

a different force, then the network modifies its output in direction of the data.

Table 2 shows to what extent the neural network method performed superior to the currently applied state-of-the-art model (cross-validated mean). The numbers indicate the relative improvement of the mean squared error of the network solution with respect to an optimally-tuned analytic model. Although the data set was very sparse and noisy, it was nevertheless still possible to give a better prediction. The effect is also shown if a different value for $\lambda$ were chosen: the higher value of $\lambda$, that is, more prior knowledge, keeps the net from memorizing the data, and improves generalization slightly. In case of the sigmoidal network, $\lambda$ was simply optimized to give the smallest cross-validation error. When trained without prior knowledge, none of the architectures lead to an improvement.

## 4   CONCLUSION

In a large-scale applications to devise a neural control for a hot line rolling mill, training data turned out to be insufficient for learning to be feasible that is only based on syntactic preference biases. By using a Bayesian framework, an imperfect domain theory was incorporated as an inductive bias in a principled way. The method outperformed the state-of-the-art solution to an extent which steelworks automation experts consider highly convincing.

## Acknowledgements

This paper describes the first two authors' joint university project, which was supported by grants from Siemens AG, Corporate R & D, and Studienstiftung des deutschen Volkes. H. Rein and F. Schmid of the Erlangen steelworks automation group helped identify the problem and sampled the data. W. Büttner and W. Finnoff made valuable suggestions.

## Footnotes

[1] This approach is related to Nowlan (1990) and MacKay (1991), but we emphasize the influence of different priors over the hypothesis space by giving preference to hypotheses that are closer to the domain theory.

[2]The normalization constants of the Gaussians in numerator and denominator cancel as well as the product for all $m \notin M(i)$ of the probabilities that $(\vec{x}_m, y_m)$ is in the data set.

[3] The costs for a single sheet of metal—giving three useful data points that have to be measured under difficult conditions—amount to a six-digit dollar sum. Only a limited number of plates of the same steel quality is processed every week, causing the data scarcity.

## References

Bergadano, F. and A. Giordana (1990). Guiding Induction with Domain Theories. In: Y. Kodratoff *et al.* (eds.), *Machine Learning*, Vol. 3, Morgan Kaufmann.

Cun, Y. Le, J. S. Denker, and S. A. Solla (1990). Optimal Brain Damage. In: D. S. Touretzky (ed.), *Advances in Neural Information Processing Systems 2*, Morgan Kaufmann.

Cun. Y. Le, I. Kanter and S. A. Solla (1991). Second Order Properties of Error Surfaces: Learning Time and Generalization. In: R. P. Lippman *et al.* (eds.), *Advances in Neural Information Processing 3*, Morgan Kaufmann.

Darken, Ch. and J. Moody (1990). Fast adaptive k-means clustering: some empirical results. In: *Proceedings of the IJCNN*, San Diego.

Duda, R. O. and P. E. Hart (1973). *Pattern Classification and Scene Analysis.* NY: Wiley.

MacKay, D. (1991). Bayesian Modeling. Ph.D. thesis, Caltech.

Minton, S. N., J. G. Carbonell *et al.* (1989). Explanation-based Learning: A problem-solving perspective. *Artificial Intelligence*, Vol. 40, pp. 63-118.

Mitchell, T. M., R. M. Keller and S. T. Kedar-Cabelli (1986). Explanation-based Learning: A unifying view. *Machine Learning*, Vol. 1, pp. 47-80.

Moody, J. (1990). Fast Learning in Multi-Resolution Hierarchies. In: D. S. Touretzky (ed.), *Advances in Neural Information Processing Systems 2*, Kaufmann, pp. 29-39.

Moody, J. and Ch. Darken (1989). Fast Learning in Networks of Locally-tuned Processing Units. *Neural Computation*, Vol. 1, pp. 281-294, MIT.

Moody, J. and Ch. Darken (1988). Learning with Localized Receptive Fields. In: D. Touretzky *et al.* (eds.), *Proc. of Connectionist Models Summer School*, Kaufmann.

Nowlan, St. J. (1990). Maximum Likelihood Competitive Learning. In: D. S. Touretzky (ed.,) *Advances in Neural Information Processing Systems 2*, Morgan Kaufmann.

Omohundro, S. M. (1991). Bump Trees for Efficient Function, Constraint, and Classification Learning. In: R. P. Lippman *et al.* (eds.), *Advances in Neural Information Processing 3*, Morgan Kaufmann.

Platt, J. (1990). A Resource-Allocating Network for Function Interpolation. In: D. S. Touretzky (ed.), *Advances in Neural Information Processing Systems 2*, Kaufmann.

Poggio, T. and F. Girosi (1990). A Theory of Networks for Approximation and Learning. A.I. Memo No. 1140 (extended in No. 1167 and No. 1253), MIT.

Röscheisen, M., R. Hofmann, and V. Tresp (1992). Incorporating Domain-Specific Prior Knowledge into Networks of Locally-Tuned Units. In: S. Hanson *et al.*(eds.), *Computational Learning Theory and Natural Learning Systems*, MIT Press.

Rumelhart, D. E., G. E. Hinton, and R. J. Williams (1986). Learning representations by back-propagating errors. *Nature*, 323(9):533-536, October.

Rumelhart, D. E. (1988). Plenary Address, IJCNN, San Diego.

Scott, G.M., J. W. Shavlik, and W. H. Ray (1991). Refining PID Controllers using Neural Networks. Technical Report, submitted to *Neural Computation*.

Tecuci, G. and Y. Kodratoff (1990). Apprenticeship Learning in Imperfect Domain Theories. In: Y. Kodratoff *et al.* (eds.), *Machine Learning*, Vol. 3, Morgan Kaufmann.

Weigend, A. (1991). Connectionist Architectures for Time-Series Prediction of Dynamical Systems. Ph.D. thesis, Stanford.